# Adaptive Access Control Applied to Ethernet Data

**Timothy X Brown**
Dept. of Electrical and Computer Engineering
University of Colorado, Boulder, CO 80309-0530
timxb@colorado.edu

## Abstract

This paper presents a method that decides which combinations of traffic can be accepted on a packet data link, so that quality of service (QoS) constraints can be met. The method uses samples of QoS results at different load conditions to build a neural network decision function. Previous similar approaches to the problem have a significant bias. This bias is likely to occur in any real system and results in accepting loads that miss QoS targets by orders of magnitude. Preprocessing the data to either remove the bias or provide a confidence level, the method was applied to sources based on difficult-to-analyze ethernet data traces. With this data, the method produces an accurate access control function that dramatically outperforms analytic alternatives. Interestingly, the results depend on throwing away more than 99% of the data.

## 1 INTRODUCTION

In a communication network in which traffic sources can be dynamically added or removed, an access controller must decide when to accept or reject a new traffic source based on whether, if added, acceptable service would be given to all carried sources. Unlike *best-effort* services such as the internet, we consider the case where traffic sources are given *quality of service* (QoS) guarantees such as maximum delay, delay variation, or loss rate. The goal of the controller is to accept the maximal number of users while guaranteeing QoS. To accommodate diverse sources such as constant bit rate voice, variable-rate video, and bursty computer data, packet-based protocols are used. We consider QOS in terms of lost packets (i.e. packets discarded due to resource overloads). This is broadly applicable (e.g. packets which violate delay guarantees can be considered lost) although some QoS measures can not fit this model.

The access control task requires a classification function—analytically or empirically derived—that specifies what conditions will result in QoS not being met. Analytic functions have been successful only on simple traffic models [Gue91], or they are so conservative that they grossly under utilize the network. This paper describes a neural network method that adapts an access control function based on historical data on what conditions packets have and have not been successfully carried. Neural based solutions have been previously applied to the access control problem [Hir90][Tra92][Est94], but these

approaches have a distinct bias that under real-world conditions leads to accepting combinations of calls that miss QoS targets by orders of magnitude. Incorporating preprocessing methods to eliminate this bias is critical and two methods from earlier work will be described. The combined data preprocessing and neural methods are applied to difficult-to-model ethernet traffic.

## 2 THE PROBLEM

Since the decision to accept a multilink connection can be decomposed into decisions on the individual links, we consider only a single link. A link can accept loads from different source types. The loads consist of packets modeled as discrete events. Arriving packets are placed in a buffer and serviced in turn. If the buffer is full, excess packets are discarded and treated as lost. The precise event timing is not critical as the concern is with the number of lost packets relative to the total number of packets received in a large sample of events, the so-called *loss rate*. The goal is to only accept load combinations which have a loss rate below the QoS target denoted by $p*$.

Load combinations are described by a feature vector, $\bar{\phi}$, consisting of load types and possibly other information such as time of day. Each feature vector, $\bar{\phi}$, has an associated loss rate, $p(\bar{\phi})$, which can not be measured directly. Therefore, the goal is to have a classifier function, $C(\bar{\phi})$, such that $C(\bar{\phi}) >, <, = 0$ if $p(\bar{\phi}) <, >, = p*$.

Since analytic $C(\bar{\phi})$ are not in general available, we look to statistical classification methods. This requires training samples, a desired output for each sample, and a significance or weight for each sample. Loads can be dynamically added or removed. Training samples are generated at load transitions, with information since the last transition containing the number of packet arrivals, $T$, the number of lost packets, $s$, and the feature vector, $\bar{\phi}$.

A sample $(\bar{\phi}_i, s_i, T_i)$, requires a desired classification, $d(\bar{\phi}_i, s_i, T_i) \in \{+1, -1\}$, and a weight, $w(\bar{\phi}_i, s_i, T_i) \in (0, \infty)$. Given a data set $\{(\bar{\phi}_i, s_i, T_i)\}$, a classifier, $C$, is then chosen that minimizes the weighted sum squared error $E = \sum_i [w(\bar{\phi}_i, s_i, T_i)(C(\bar{\phi}_i) - d(\bar{\phi}_i, s_i, T_i))^2]$.

A classifier, with enough degrees of freedom will set $C(\bar{\phi}_i) = d(\bar{\phi}_i, s_i, T_i)$ if all the $\bar{\phi}_i$ are different. With multiple samples at the same $\bar{\phi}$ then we see that the error is minimized when

$$C(\bar{\phi}) = \left(\sum_{\{i|\bar{\phi}_i = \bar{\phi}\}} [w(\bar{\phi}_i, s_i, T_i)d(\bar{\phi}_i, s_i, T_i)]\right) \Big/ \left(\sum_{\{i|\bar{\phi}_i = \bar{\phi}\}} w(\bar{\phi}_i, s_i, T_i)\right). \quad (1)$$

Thus, the optimal $C(\bar{\phi})$ is the weighted average of the $d(\bar{\phi}_i, s_i, T_i)$ at $\bar{\phi}$. If the classifier has fewer degrees of freedom (e.g. a low dimension linear classifier), $C(\bar{\phi})$ will be the average of the $d(\bar{\phi}_i, s_i, T_i)$ in the neighborhood of $\bar{\phi}$, where the neighborhood is, in general, an unspecified function of the classifier.

A more direct form of averaging would be to choose a specific neighborhood around $\bar{\phi}$ and average over samples in this neighborhood. This suffers from having to store all the samples in the decision mechanism, and incurs a significant computational burden. More significant is how to decide the size of the neighborhood. If it is fixed, in sparse regions no samples may be in the neighborhood. In dense regions near decision boundaries, it may average over too wide a range for accurate estimates. Dynamically setting the neighborhood so that it always contains the $k$ nearest neighbors solves this problem, but does not account for the size of the samples. We will return to this in Section 4.

## 3 THE SMALL SAMPLE PROBLEM

Neural networks have previously been applied to the access control problem [Hir91] [Tra92][Est94]. In [Hir90] and [Tra92], $d(\bar{\phi}_i, s_i, T_i) = +1$ when $s_i/T_i < p*$, $d(\bar{\phi}_i, s_i, T_i) = -1$ otherwise, and the weighting is a uniform $w(\bar{\phi}_i, s_i, T_i) = 1$ for all $i$. This desired out and

uniform weighting we call the *normal* method. For a given load combination, $\bar{\phi}$, assume an idealized system where packets enter and with probability $p(\bar{\phi})$ independent of earlier or later packets, the packet is labeled as lost. In a sample of $T$ such Bernoulli trials with $s$ the number packets lost, let $P_B = P\{s/T > p*\}$. Since with the normal method $d(\bar{\phi}, s, T) = -1$ if $s/T > p*$, $P_B = P\{d(\bar{\phi}, s, T) = -1\}$. From (1), with uniform weighting the decision boundary is where $P_B = 0.5$. If the samples are small (i.e. $T < (\ln 2)/p* < 1/p*$), $d(\bar{\phi}, s, T) = -1$ for all $s > 0$. In this case $P_B = 1 - (1 - p(\bar{\phi}))^T$. Solving for $p(\bar{\phi})$ at $P_B = 0.5$ using $\ln(1 - x) \approx -x$, the decision boundary is at $p(\bar{\phi}) \approx (\ln 2)/T > p*$. So, for small sample sizes, the normal method boundary is biased to greater than $p*$ and can be made orders of magnitude larger as $T$ becomes smaller. For larger $T$, e.g. $Tp* > 10$, this bias will be seen to be negligible.

One obvious solution is to have large samples. This is complicated by three effects. The first is that desired loss rates in data systems are often small; typically in the range $10^{-6}$–$10^{-12}$. This implies that to be large, samples must be at least $10^7$–$10^{13}$ packets. For the latter, even at Gbps rates, short packets, and full loading this translates into samples of several hours of traffic. Even for the first at typical rates, this can translate into minutes of traffic. The second, related problem is that in dynamic data networks, while individual connections may last for significant periods, on the aggregate a given combination of loads may not exist for the requisite period. The third more subtle problem is that in any queueing system even with uncorrelated arrival traffic the buffering introduces memory in the system. A typical sample with losses may contain 100 losses, but a loss trace would show that all of the losses occurred in a single short overload interval. Thus the number of independent trials can be several orders of magnitude smaller than indicated by the raw sample size indicating that the loads must be stable for hours, days, or even years to get samples that lead to unbiased classification.

An alternative approach used in [Hir95] sets $d(\bar{\phi}, s, T) = s/T$ and models $p(\bar{\phi})$ directly. The probabilities can vary over orders of magnitude making accurate estimates difficult. Estimating the less variable $\log(p(\bar{\phi}))$ with $d = \log(s/T)$ is complicated by the logarithm being undefined for small samples where most samples have no losses so that $s = 0$.

## 4   METHODS FOR TREATING BIAS AND VARIANCE

We present without proof two preprocessing methods derived and analyzed in [Bro96]. The first eliminates the sample bias by choosing an appropriate $d$ and $w$ that directly solves (1) s.t. $C(\bar{\phi}) >, <, = 0$ if and only if $p(\bar{\phi}) <, >, = p*$ i.e. it is an unbiased estimate as to whether the loss rate is above and below $p*$. This is the *weighting* method shown in Table 1. The relative weighting of samples with loss rates above and below the critical loss rate is plotted in Figure 1. For large $T$, as expected, it reduces to the normal method.

The second preprocessing method assigns uniform weighting, but classifies $d(\bar{\phi}, s, T) = 1$ only if a certain confidence level, $L$, is met that the sample represents a combination where $p(\bar{\phi}) < p*$. Such a confidence was derived in [Bro96]:

Table 1: Summary of Methods.

| Method | Sample Class $d(\bar{\phi}_i, s_i, T_i) = +1$ if | Weighting, $w(\bar{\phi}_i, s_i, T_i)$, when | |
|---|---|---|---|
| | | $d(\bar{\phi}_i, s_i, T_i) = +1$ (i.e. $w^+$) | $d(\bar{\phi}_i, s_i, T_i) = -1$ (i.e. $w^-$) |
| Normal | $s_i \le \lfloor p*T \rfloor$ | 1 | 1 |
| Weighting | $s_i \le \lfloor p*T \rfloor$ | $T\displaystyle\sum_{i > \lfloor p*T \rfloor} \binom{T}{i} p*^i (1 - p*)^{T-i}$ | $T\displaystyle\sum_{i \le \lfloor p*T \rfloor} \binom{T}{i} p*^i (1 - p*)^{T-i}$ |
| Aggregate | Table 2 | 1 | 1 |

$$P\{p(\bar{\phi}) > p* \,|\, s, T\} \cong e^{-Tp*} \sum_{i=0}^{s} \frac{(Tp*)^i}{i!} \tag{2}$$

For small $T$ (e.g. $T < 1/p*$ and $L > 1 - 1/e$), even if $s = 0$ (no losses), this level is not met. But, a neighborhood of samples with similar load combinations may all have no losses indicating that this sample can be classified as having $p(\bar{\phi}) < p*$. Choosing a neighborhood requires a metric, $m$, between feature vectors, $\bar{\phi}$. In this paper we simply use Euclidean distance. Using the above and solving for $T$ when $s = 0$, the smallest meaningful neighborhood size is the smallest $k$ such that the aggregate sample is greater than a critical size, $T* = -\ln(1 - L)/p*$. From (2), this guarantees that if no packets in the aggregate sample are lost we can classify it as having $p(\bar{\phi}) < p*$ within our confidence level. For larger samples, or where samples are more plentiful and $k$ can afford to be large, (2) can be used directly. Table 2 summarizes this *aggregate* method.

The above preprocessing methods assume that the training samples consist of independent samples of Bernoulli trials. Because of memory introduced by the buffer and possible correlations in the arrivals, this is decidedly not true. The methods can still be applied, if samples can be subsampled at every $l$th trial where $l$ is large enough so that the samples are pseudo-independent, i.e. the dependency is not significant for our application.

A simple graphical method for determining $l$ is as follows. Observing Figure 1, if the number of trials is artificially increased, for small samples the weighting method will tend to under weight the trials with errors, so that its decision boundary will be at erroneously high loss rates. This is the case with correlated samples. The sample size, $T$, overstates the number of independent trials. As the subsample factor is increased, the subsample size becomes smaller, the trials become increasingly independent, the weighting becomes more appropriate, and the decision boundary moves closer to the true decision boundary. At some point, the samples are sufficiently independent so that sparser subsampling does not change the decision boundary. By plotting the decision boundary of the classifier as a function of $l$, the point where the boundary is independent of the subsample factor indicates a suitable choice for $l$.

In summary, the procedure consists of collecting traffic samples at different combinations of traffic loads that do and do not meet quality of service. These are then subsampled with a factor $l$ determined as above. Then one of the sample preprocessing methods, summarized in Table 1, are applied to the data. These preprocessed samples are then used in any neural network or classification scheme. Analysis in [Bro96] derives the expected bias (shown in Figure 2) of the methods when used with an ideal classifier. The normal method can be arbitrarily biased, the weighting method is unbiased, and the aggregate method chooses a conservative boundary. Simulation experiments in [Bro96] applying it to a well characterized M/M/1 queueing system to determine acceptable loads showed that the weighting method was able to produce unbiased threshold estimates over a range of val-

Table 2: Aggregate Classification Algorithm

| |
|---|
| 1. Given Sample $(\bar{\phi}_i, s_i, T_i) \in \{(\bar{\phi}_i, s_i, T_i)\}$, metric, $m$, and confidence level, $L$. |
| 2. Calculate $T* = -\ln(1 - L)/p*$. |
| 3. Find nearest neighbors $n_0, n_1, \ldots$ where $n_0 = i$ and $m(\bar{\phi}_{n_j}, \bar{\phi}_i) \le m(\bar{\phi}_{n_{j+1}}, \bar{\phi}_i)$ for $j \ge 0$. |
| 4. Choose smallest $k$ s.t. $T' = \displaystyle\sum_{j=0}^{k} T_{n_j} \ge T*$. Let $s' = \displaystyle\sum_{j=0}^{k} s_{n_j}$. |
| 5. Using (2), $d(\bar{\phi}_i, s_i, T_i) = \begin{cases} +1 & \text{if } P\{p(\bar{\phi}) > p* \,|\, s', T'\} < (1 - L) \\ -1 & \text{o.w.} \end{cases}$. |

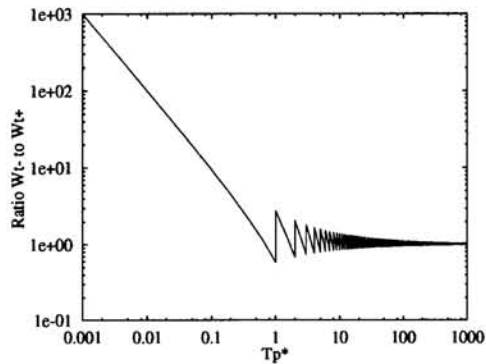

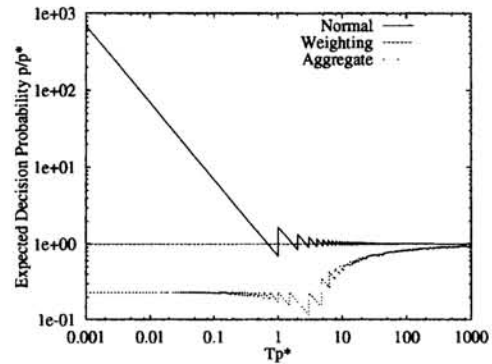

Figure 1: Plot of Relative Weighting of Samples with Losses Below $(w^-)$ and Above $(w^+)$ the Critical Loss Rate.

Figure 2: Expected Decision Normalized by $p*$. The nominal boundary is $p/p* = 1$. The aggregate method uses $L = 0.95$.

ues; and the aggregate method produced conservative estimates that were always below the desired threshold, although in terms of traffic load were only 5% smaller. Even in this simple system where the input traffic is uncorrelated (but the losses become correlated due the memory in the queue), the subsample factor was 12, meaning that good results required more than 90% of the data be thrown out.

## 5   EXPERIMENTS WITH ETHERNET TRAFFIC DATA

This paper set out to solve the problem of access control for real world data. We consider a system where the call combinations consist of individual computer data users trunked onto a single output link. This is modeled as a discrete-time single-server queueing model where in each time slot one packet can be processed and zero or more packets can arrive from the different users. The server has a buffer of fixed length 1000. To generate a realistic arrival process, we use ethernet data traces. The bandwidth of the link was chosen at from 10–100Mbps. With 48 byte packets, the queue packet service rate was the bandwidth divided by 384. All arrival rates are normalized by the service rate.

### 5.1   THE DATA

We used ethernet data described in [Lel93] as the August 89 busy hour containing traffic ranging from busy file-servers/routers to users with just a handful of packets. The detailed data set records every packet's arrival time (to the nearest 100μsec), size, plus source and destination tags. From this, 108 "data traffic" sources were generated, one for each computer that generated traffic on the ethernet link. To produce uniform size packets, each ethernet packet (which ranged from 64 to 1518 bytes) was split into 2 to 32 48-byte packets (partial packets were padded to 48 bytes). Each ethernet packet arrival time was mapped into a particular time slot in the queueing model. All the packets arriving in a timeslot are immediately added to the buffer, any buffer overflows would be discarded (counted as lost), and if the buffer was non-empty at the start of the timeslot, one packet sent. Ethernet contains a collision protocol so that only one of the sources is sending packets at any one time onto a 10Mbps connection. Decorrelating the sources via random starting offsets, produced independent data sources with the potential for overloads. Multiple copies at different offsets produced sufficient loads even for bandwidths greater than 10Mbps.

The peak data rate with this data is fixed, while the load (the average rate over the one hour trace normalized by the peak rate) ranges over five orders of magnitude. Also troubling, analysis of this data [Lel93] shows that the aggregate traffic exhibits chaotic self-similar properties and suggests that it may be due to the sources' distribution of packet inter-arrival times following an extremely heavy tailed distribution with infinite higher order moments. No tractable closed form solution exists for such data to predict whether a particular load will result in an overload. Thus, we apply adaptive access control.

## 5.2 EXPERIMENT AND RESULTS

We divided the data into two roughly similar groups of 54 sources each; one for training and one for testing. To create sample combinations we assign a distribution over the different training sources, choose a source combination from this distribution, and choose a random, uniform (over the period of the trace) starting time for each source. Simulations that reach the end of a trace wrap around to the beginning of the trace. The sources are described by a single feature corresponding to the average load of the source over the one hour data trace. A group of sources is described by the sum of the average loads. The source distribution was a uniformly chosen $0-M$ copies of each of the 54 training samples. $M$ was dynamically chosen so that the link would be sufficiently loaded to cause losses. Each sample combination was processed for $3 \times 10^7$ time slots, recording the load combination, the number of packets serviced correctly, and the number blocked. The experiment was repeated for a range of bandwidths. The bandwidths and number of samples at each bandwidth are shown in Table 3

We applied the three methods of Table 1 based on $p* = 10^{-6}$ ($L = 95\%$ for the aggregate method) and used the resulting data in a linear classifier. Since the feature is the load and larger loads will always cause more blocking, $p(\phi)$ is a one variable monotonic function. A linear classifier is sufficient for this case and its output is simply a threshold on the load.

To create pseudo-independent trials necessary for the aggregate and weighting methods, we subsampled every $I$th packet. Using the graphical method of Section 4, the resulting $I$ are shown in column 4 of Table 3. A typical subsample factor is 200. The sample sizes ranged from $10^5$ to $10^7$ trials, But, after subsampling by a factor of 200, even for the largest samples, $p*T < 0.05 \ll 1$.

The thresholds found by each method are shown in Table 3. To get loss rate estimates at these thresholds, the average loss rate of the 20% of source combinations below each method's threshold is computed. Since accepted loads would be below the threshold this is a typical loss rate. The normal scheme is clearly flawed with losses 10 times higher than $p*$, the weighting scheme's loss rate is apparently unbiased with results around $p*$, while the aggregate scheme develops a conservative boundary below $p*$. To test the boundaries, we repeated the experiment generating source combination samples using the 54 sources not used in the training. Table 3 also shows the losses on this test set and indicates that the training set boundaries produce similar results on the test data.

The boundaries are compared with those of more conventional, model-based techniques. One proposed technique for detecting overloads appears in [Gue91]. This paper assumes the sources are based on a Markov On/Off model. Applying the method to this ethernet data (treating each packet arrival as an On period and calculating necessary parameters from there), all but the very highest loads in the training sets are classified as acceptable indicating that the loss rate would be orders of magnitude higher than $p*$. A conservative technique is to accept calls only as long as the sum of the peak source transmission rates is less than the link bandwidth. For the 10Mbps link, since this equals the original ethernet

Table 3: Results from Experiments at Different Link Bandwidth.

| Band-width (Mbps) | Number of Samples | | Sub-sample Factor | Threshold Found & Loss Rate at Threshold on (train/test) Set | | |
|---|---|---|---|---|---|---|
| | Train | Test | | Normal | Weighting | Aggregate |
| 10 | 1569 | 1080 | 230 | 0.232 (1e–5/4e–6) | 0.139 (8e–7/1e–6) | 0.105 (1e–7/8e–8) |
| 17.5 | 2447 | 3724 | 180 | 0.415 (2e–5/3e–5) | 0.268 (5e–7/9e–7) | 0.215 (3e–9/4e–7) |
| 30 | 6696 | 4219 | 230 | 0.508 (7e–6/4e–5) | 0.333 (4e–6/5e–8) | 0.286 (3e–7/2e–8) |
| 100 | 1862 | N.A. | 180 | 0.688 (1e–5/N.A.) | 0.566 (5e–7/N.A.) | 0.494 (0e–0/N.A.) |

data rate, this peak rate method will accept exactly one source. Averaging over all sources, the average load would be 0.0014 and would not increase with increasing bandwidth. The neural method takes advantage of better trunking at increasing bandwidths, and carries two orders of magnitude more traffic.

# 6  CONCLUSION

Access control depends on a classification function that decides if a given set of load conditions will violate quality of service constraints. In this paper quality of service was in terms of a maximum packet loss rate, $p*$. Given that analytic methods are inadequate when given realistic traffic sources, a neural network classification method based on samples of traffic results at different load conditions is a practical alternative. With previous neural network approaches, the synthetic nature of the experiments obscured a significant bias that exists with more realistic data. This bias, due to the small sample sizes relative to $1/p*$, is likely to occur in any real system and results in accepting loads with losses that are orders of magnitude greater than $p*$.

Preprocessing the data to either remove the bias or provide a confidence level, the neural network was applied to sources based on difficult-to-analyze ethernet data traces. A group of sources was characterized by its total load so that the goal was to simply choose a threshold on how much load the link would accept. The neural network was shown to produce accurate estimates of the correct threshold. Interestingly these good results depend on creating traffic samples representing independent packet transmissions. This requires more than 99% of the data to be thrown away indicating that for good performance an easy-to-implement sparse sampling of the packet fates is sufficient. It also indicates that unless the total number of packets that is observed is orders of magnitude larger than $1/p*$, the samples are actually small and preprocessing methods such as in this paper must be applied for accurate loss rate classification.

In comparison to analytic techniques, all of the methods, are more accurate at identifying overloads. In comparison to the best safe alternative that works even on this ethernet data, the neural network method was able to carry two orders of magnitude more traffic. The techniques in this paper apply to a range of network problems from routing, to bandwidth allocation, network design, as well as access control.

**References**

[Bro96] Brown, T.X, "Classifying Loss Rates with Small Samples," Submitted to *IEEE Tran. on Comm.*, April 1996.

[Est94] Estrella, A.D., Jurado, A., Sandoval, F., "New Training Pattern Selection Method for ATM Call Admission Neural Control," *Elec. Let.*, Vol. 30, No. 7, pp. 577–579, Mar. 1994.

[Gue91] Guerin, R., Ahmadi, H., Naghshineh, M., "Equivalent Capacity and its Application to Bandwidth Allocation in High-Speed Networks," *IEEE JSAC*, vol. 9, no. 7, pp. 968–981, 1991.

[Hir90] Hiramatsu, A., "ATM Communications Network Control by Neural Networks," *IEEE Trans. on Neural Networks*, vol. 1, no. 1, pp. 122–130, 1990.

[Hir95] Hiramatsu, A., "Training Techniques for Neural Network Applications in ATM," *IEEE Comm. Mag.*, October, pp. 58–67, 1995.

[Lel93] Leland, W.E., Taqqu, M.S., Willinger, W., Wilson, D.V., "On the Self-Similar Nature of Ethernet Traffic," in *Proc. of ACM SIGCOMM* 1993. pp. 183–193.

[Tra92] Tran-Gia, P., Gropp, O., "Performance of a Neural Net used as Admission Controller in ATM Systems," *Proc. Globecom 92*, Orlando, FL, pp. 1303–1309.
